# Transduction with Matrix Completion: Three Birds with One Stone

**Andrew B. Goldberg**[1], **Xiaojin Zhu**[1], **Benjamin Recht**[1], **Jun-Ming Xu**[1], **Robert Nowak**[2]
Department of {[1]Computer Sciences, [2]Electrical and Computer Engineering}
University of Wisconsin-Madison, Madison, WI 53706
{goldberg, jerryzhu, brecht, xujm}@cs.wisc.edu, nowak@ece.wisc.edu

## Abstract

We pose transductive classification as a matrix completion problem. By assuming the underlying matrix has a low rank, our formulation is able to handle three problems simultaneously: i) multi-label learning, where each item has more than one label, ii) transduction, where most of these labels are unspecified, and iii) missing data, where a large number of features are missing. We obtained satisfactory results on several real-world tasks, suggesting that the low rank assumption may not be as restrictive as it seems. Our method allows for different loss functions to apply on the feature and label entries of the matrix. The resulting nuclear norm minimization problem is solved with a modified fixed-point continuation method that is guaranteed to find the global optimum.

## 1   Introduction

Semi-supervised learning methods make assumptions about how unlabeled data can help in the learning process, such as the manifold assumption (data lies on a low-dimensional manifold) and the cluster assumption (classes are separated by low density regions) [4, 16]. In this work, we present two transductive learning methods under the novel assumption that the feature-by-item and label-by-item matrices are *jointly low rank*. This assumption effectively couples different label prediction tasks, allowing us to implicitly use observed labels in one task to recover unobserved labels in others. The same is true for imputing missing features. In fact, our methods learn in the difficult regime of *multi-label transductive learning with missing data* that one sometimes encounters in practice. That is, each item is associated with many class labels, many of the items' labels may be unobserved (some items may be completely unlabeled across all labels), and many features may also be unobserved. Our methods build upon recent advances in matrix completion, with efficient algorithms to handle matrices with mixed real-valued features and discrete labels. We obtain promising experimental results on a range of synthetic and real-world data.

## 2   Problem Formulation

Let $\mathbf{x}_1 \ldots \mathbf{x}_n \in \mathbb{R}^d$ be feature vectors associated with $n$ items. Let $\mathbf{X} = [\mathbf{x}_1 \ldots \mathbf{x}_n]$ be the $d \times n$ feature matrix whose columns are the items. Let there be $t$ binary classification tasks, $\mathbf{y}_1 \ldots \mathbf{y}_n \in \{-1, 1\}^t$ be the label vectors, and $\mathbf{Y} = [\mathbf{y}_1 \ldots \mathbf{y}_n]$ be the $t \times n$ label matrix. Entries in $\mathbf{X}$ or $\mathbf{Y}$ can be missing at random. Let $\Omega_{\mathbf{X}}$ be the index set of observed features in $\mathbf{X}$, such that $(i, j) \in \Omega_{\mathbf{X}}$ if and only if $x_{ij}$ is observed. Similarly, let $\Omega_{\mathbf{Y}}$ be the index set of observed labels in $\mathbf{Y}$. Our main goal is to predict the missing labels $y_{ij}$ for $(i, j) \notin \Omega_{\mathbf{Y}}$. Of course, this reduces to standard transductive learning when $t = 1$, $|\Omega_{\mathbf{X}}| = nd$ (no missing features), and $1 < |\Omega_{\mathbf{Y}}| < n$ (some missing labels). In our more general setting, as a side product we are also interested in imputing the missing features, and de-noising the observed features, in $\mathbf{X}$.

## 2.1 Model Assumptions

The above problem is in general ill-posed. We now describe our assumptions to make it a well-defined problem. In a nutshell, we assume that $\mathbf{X}$ and $\mathbf{Y}$ are jointly produced by an underlying low rank matrix. We then take advantage of the sparsity to fill in the missing labels and features using a modified method of matrix completion. Specifically, we assume the following generative story. It starts from a $d \times n$ low rank "pre"-feature matrix $\mathbf{X^0}$, with $\mathrm{rank}(\mathbf{X^0}) \ll \min(d, n)$. The actual feature matrix $\mathbf{X}$ is obtained by adding *iid* Gaussian noise to the entries of $\mathbf{X^0}$: $\mathbf{X} = \mathbf{X^0} + \epsilon$, where $\epsilon_{ij} \sim N(0, \sigma_\epsilon^2)$. Meanwhile, the $t$ "soft" labels $\left(y_{1j}^0 \ldots y_{tj}^0\right)^\top \equiv \mathbf{y}_j^0 \in \mathbb{R}^t$ of item $j$ are produced by $\mathbf{y}_j^0 = \mathbf{W}\mathbf{x}_j^0 + \mathbf{b}$, where $\mathbf{W}$ is a $t \times d$ weight matrix, and $\mathbf{b} \in \mathbb{R}^t$ is a bias vector. Let $\mathbf{Y^0} = \left[\mathbf{y}_1^0 \ldots \mathbf{y}_n^0\right]$ be the soft label matrix. Note the combined $(t + d) \times n$ matrix $\left[\mathbf{Y^0}; \mathbf{X^0}\right]$ is low rank too: $\mathrm{rank}(\left[\mathbf{Y^0}; \mathbf{X^0}\right]) \leq \mathrm{rank}(\mathbf{X^0}) + 1$. The actual label $y_{ij} \in \{-1, 1\}$ is generated randomly via a sigmoid function: $P(y_{ij}|y_{ij}^0) = 1/\left(1 + \exp(-y_{ij}y_{ij}^0)\right)$. Finally, two random masks $\Omega_\mathbf{X}, \Omega_\mathbf{Y}$ are applied to expose only some of the entries in $\mathbf{X}$ and $\mathbf{Y}$, and we use $\omega$ to denote the percentage of observed entries. This generative story may seem restrictive, but our approaches based on it perform well on synthetic and real datasets, outperforming several baselines with linear classifiers.

## 2.2 Matrix Completion for Heterogeneous Matrix Entries

With the above data generation model, our task can be defined as follows. Given the partially observed features and labels as specified by $\mathbf{X}, \mathbf{Y}, \Omega_\mathbf{X}, \Omega_\mathbf{Y}$, we would like to recover the intermediate low rank matrix $\left[\mathbf{Y^0}; \mathbf{X^0}\right]$. Then, $\mathbf{X^0}$ will contain the denoised and completed features, and $\mathrm{sign}(\mathbf{Y^0})$ will contain the completed and correct labels.

The key assumption is that the $(t + d) \times n$ stacked matrix $\left[\mathbf{Y^0}; \mathbf{X^0}\right]$ is of low rank. We will start from a "hard" formulation that is illustrative but impractical, then relax it.

$$\underset{\mathbf{Z} \in \mathbb{R}^{(t+d) \times n}}{\mathrm{argmin}} \quad \mathrm{rank}(\mathbf{Z}) \tag{1}$$

$$\text{s.t.} \quad \mathrm{sign}(z_{ij}) = y_{ij}, \ \ \forall (i,j) \in \Omega_\mathbf{Y}; \qquad z_{(i+t)j} = x_{ij}, \ \ \forall(i,j) \in \Omega_\mathbf{X}$$

Here, $\mathbf{Z}$ is meant to recover $\left[\mathbf{Y^0}; \mathbf{X^0}\right]$ by directly minimizing the rank while obeying the observed features and labels. Note the indices $(i, j) \in \Omega_\mathbf{X}$ are with respect to $\mathbf{X}$, such that $i \in \{1, \ldots, d\}$. To index the corresponding element in the larger stacked matrix $\mathbf{Z}$, we need to shift the row index by $t$ to skip the part for $\mathbf{Y^0}$, and hence the constraints $z_{(i+t)j} = x_{ij}$. The above formulation assumes that there is no noise in the generation processes $\mathbf{X^0} \to \mathbf{X}$ and $\mathbf{Y^0} \to \mathbf{Y}$. Of course, there are several issues with formulation (1), and we handle them as follows:

- $\mathrm{rank}()$ is a non-convex function and difficult to optimize. Following recent work in matrix completion [3, 2], we relax $\mathrm{rank}()$ with the convex nuclear norm $\|\mathbf{Z}\|_* = \sum_{k=1}^{\min(t+d,n)} \sigma_k(\mathbf{Z})$, where $\sigma_k$'s are the singular values of $\mathbf{Z}$. The relationship between $\mathrm{rank}(\mathbf{Z})$ and $\|\mathbf{Z}\|_*$ is analogous to that of $\ell^0$-norm and $\ell^1$-norm for vectors.

- There is feature noise from $\mathbf{X^0}$ to $\mathbf{X}$. Instead of the equality constraints in (1), we minimize a loss function $c_x(z_{(i+t)j}, x_{ij})$. We choose the squared loss $c_x(u, v) = \frac{1}{2}(u - v)^2$ in this work, but other convex loss functions are possible too.

- Similarly, there is label noise from $\mathbf{Y^0}$ to $\mathbf{Y}$. The observed labels are of a different type than the observed features. We therefore introduce another loss function $c_y(z_{ij}, y_{ij})$ to account for the heterogeneous data. In this work, we use the logistic loss $c_y(u, v) = \log(1 + \exp(-uv))$.

In addition to these changes, we will model the bias $\mathbf{b}$ either explicitly or implicitly, leading to two alternative matrix completion formulations below.

**Formulation 1 (MC-b).** In this formulation, we explicitly optimize the bias $\mathbf{b} \in \mathbb{R}^t$ in addition to $\mathbf{Z} \in \mathbb{R}^{(t+d) \times n}$, hence the name. Here, $\mathbf{Z}$ corresponds to the stacked matrix $\left[\mathbf{W}\mathbf{X}^0; \mathbf{X}^0\right]$ instead of $\left[\mathbf{Y^0}; \mathbf{X^0}\right]$, making it potentially lower rank. The optimization problem is

$$\underset{\mathbf{Z}, \mathbf{b}}{\mathrm{argmin}} \quad \mu\|\mathbf{Z}\|_* + \frac{\lambda}{|\Omega_\mathbf{Y}|} \sum_{(i,j) \in \Omega_\mathbf{Y}} c_y(z_{ij} + b_i, y_{ij}) + \frac{1}{|\Omega_\mathbf{X}|} \sum_{(i,j) \in \Omega_\mathbf{X}} c_x(z_{(i+t)j}, x_{ij}), \tag{2}$$

where $\mu, \lambda$ are positive trade-off weights. Notice the bias $\mathbf{b}$ is not regularized. This is a convex problem, whose optimization procedure will be discussed in section 3. Once the optimal $\mathbf{Z}, \mathbf{b}$ are found, we recover the task-$i$ label of item $j$ by $\text{sign}(z_{ij} + b_i)$, and feature $k$ of item $j$ by $z_{(k+t)j}$.

**Formulation 2 (MC-1).** In this formulation, the bias is modeled implicitly within $\mathbf{Z}$. Similar to how bias is commonly handled in linear classifiers, we append an additional feature with constant value one to each item. The corresponding pre-feature matrix is augmented into $\left[ \mathbf{X^0}; \mathbf{1}^\top \right]$, where $\mathbf{1}$ is the all-1 vector. Under the same label assumption $\mathbf{y}_j^0 = \mathbf{W} \mathbf{x}_j^0 + \mathbf{b}$, the rows of the soft label matrix $\mathbf{Y^0}$ are linear combinations of rows in $\left[ \mathbf{X^0}; \mathbf{1}^\top \right]$, i.e., $\text{rank}(\left[ \mathbf{Y^0}; \mathbf{X^0}; \mathbf{1}^\top \right]) = \text{rank}(\left[ \mathbf{X^0}; \mathbf{1}^\top \right])$. We then let $\mathbf{Z}$ correspond to the $(t + d + 1) \times n$ stacked matrix $\left[ \mathbf{Y^0}; \mathbf{X^0}; \mathbf{1}^\top \right]$, by forcing its last row to be $\mathbf{1}^\top$ (hence the name):

$$\operatorname*{argmin}_{\mathbf{Z} \in \mathbb{R}^{(t+d+1) \times n}} \quad \mu \| \mathbf{Z} \|_* + \frac{\lambda}{|\Omega_{\mathbf{Y}}|} \sum_{(i,j) \in \Omega_{\mathbf{Y}}} c_y(z_{ij}, y_{ij}) + \frac{1}{|\Omega_{\mathbf{X}}|} \sum_{(i,j) \in \Omega_{\mathbf{X}}} c_x(z_{(i+t)j}, x_{ij}) \quad (3)$$

$$\text{s.t.} \quad z_{(t+d+1)\cdot} = \mathbf{1}^\top.$$

This is a constrained convex optimization problem. Once the optimal $\mathbf{Z}$ is found, we recover the task-$i$ label of item $j$ by $\text{sign}(z_{ij})$, and feature $k$ of item $j$ by $z_{(k+t)j}$.

MC-b and MC-1 differ mainly in what is in $\mathbf{Z}$, which leads to different behaviors of the nuclear norm. Despite the generative story, we do not explicitly recover the weight matrix $\mathbf{W}$ in these formulations. Other formulations are certainly possible. One way is to let $\mathbf{Z}$ correspond to $\left[ \mathbf{Y^0}; \mathbf{X^0} \right]$ directly, without introducing bias $\mathbf{b}$ or the all-1 row, and hope nuclear norm minimization will prevail. This is inferior in our preliminary experiments, and we do not explore it further in this paper.

# 3 Optimization Techniques

We solve MC-b and MC-1 using modifications of the Fixed Point Continuation (FPC) method of Ma, Goldfarb, and Chen [10].[1] While nuclear norm minimization can be converted into a semidefinite programming (SDP) problem [2], current SDP solvers are severely limited in the size of problems they can solve. Instead, the basic fixed point approach is a computationally efficient alternative, which provably converges to the globally optimal solution and has been shown to outperform SDP solvers in terms of matrix recoverability.

## 3.1 Fixed Point Continuation for MC-b

We first describe our modified FPC method for MC-b. It differs from [10] in the extra bias variables and multiple loss functions. Our fixed point iterative algorithm to solve the unconstrained problem of (2) consists of two alternating steps for each iteration $k$:

1. (gradient step) $\mathbf{b}^{k+1} = \mathbf{b}^k - \tau_{\mathbf{b}} g(\mathbf{b}^k)$, $\mathbf{A}^k = \mathbf{Z}^k - \tau_{\mathbf{Z}} g(\mathbf{Z}^k)$
2. (shrinkage step) $\mathbf{Z}^{k+1} = S_{\tau_{\mathbf{Z}} \mu}(\mathbf{A}^k)$.

In the gradient step, $\tau_{\mathbf{b}}$ and $\tau_{\mathbf{Z}}$ are step sizes whose choice will be discussed next. Overloading notation a bit, $g(\mathbf{b}^k)$ is the vector gradient, and $g(\mathbf{Z}^k)$ is the matrix gradient, respectively, of the two loss terms in (2) (i.e., excluding the nuclear norm term):

$$g(b_i) = \frac{\lambda}{|\Omega_{\mathbf{Y}}|} \sum_{j:(i,j) \in \Omega_{\mathbf{Y}}} \frac{-y_{ij}}{1 + \exp(y_{ij}(z_{ij} + b_i))} \quad (4)$$

$$g(z_{ij}) = \begin{cases} \frac{\lambda}{|\Omega_{\mathbf{Y}}|} \frac{-y_{ij}}{1 + \exp(y_{ij}(z_{ij} + b_i))}, & i \leq t \text{ and } (i,j) \in \Omega_{\mathbf{Y}} \\ \frac{1}{|\Omega_{\mathbf{X}}|}(z_{ij} - x_{(i-t)j}), & i > t \text{ and } (i - t, j) \in \Omega_{\mathbf{X}} \\ 0, & \text{otherwise} \end{cases} \quad (5)$$

Note for $g(z_{ij}), i > t$, we need to shift down (un-stack) the row index by $t$ in order to map the element in $\mathbf{Z}$ back to the item $x_{(i-t)j}$.

**Input**: Initial matrix $\mathbf{Z}_0$, bias $\mathbf{b}_0$,
          parameters $\mu, \lambda$, Step sizes $\tau_{\mathbf{b}}, \tau_{\mathbf{Z}}$
Determine $\mu_1 > \mu_2 > \cdots > \mu_L = \mu > 0$.
Set $\mathbf{Z} = \mathbf{Z}_0, \mathbf{b} = \mathbf{b}_0$.
**foreach** $\mu = \mu_1, \mu_2, \ldots, \mu_L$ **do**
    **while** *Not converged* **do**
        Compute $\mathbf{b} = \mathbf{b} - \tau_{\mathbf{b}} g(\mathbf{b}), \mathbf{A} = \mathbf{Z} - \tau_{\mathbf{Z}} g(\mathbf{Z})$
        Compute SVD of $\mathbf{A} = \mathbf{U}\Lambda\mathbf{V}^{\top}$
        Compute $\mathbf{Z} = \mathbf{U}\max(\Lambda - \tau_{\mathbf{Z}}\mu, 0)\mathbf{V}^{\top}$
    **end**
**end**
**Output**: Recovered matrix $\mathbf{Z}$, bias $\mathbf{b}$

**Algorithm 1**: FPC algorithm for MC-b.

**Input**: Initial matrix $\mathbf{Z}_0$,
          parameters $\mu, \lambda$, Step sizes $\tau_{\mathbf{Z}}$
Determine $\mu_1 > \mu_2 > \cdots > \mu_L = \mu > 0$.
Set $\mathbf{Z} = \mathbf{Z}_0$.
**foreach** $\mu = \mu_1, \mu_2, \ldots, \mu_L$ **do**
    **while** *Not converged* **do**
        Compute $\mathbf{A} = \mathbf{Z} - \tau_{\mathbf{Z}} g(\mathbf{Z})$
        Compute SVD of $\mathbf{A} = \mathbf{U}\Lambda\mathbf{V}^{\top}$
        Compute $\mathbf{Z} = \mathbf{U}\max(\Lambda - \tau_{\mathbf{Z}}\mu, 0)\mathbf{V}^{\top}$
        Project $\mathbf{Z}$ to feasible region $z_{(t+d+1)\cdot} = \mathbf{1}^{\top}$
    **end**
**end**
**Output**: Recovered matrix $\mathbf{Z}$

**Algorithm 2**: FPC algorithm for MC-1.

In the shrinkage step, $S_{\tau_{\mathbf{Z}}\mu}(\cdot)$ is a matrix shrinkage operator. Let $\mathbf{A}^k = \mathbf{U}\Lambda\mathbf{V}^{\top}$ be the SVD of $\mathbf{A}^k$. Then $S_{\tau_{\mathbf{Z}}\mu}(\mathbf{A}^k) = \mathbf{U}\max(\Lambda - \tau_{\mathbf{Z}}\mu, 0)\mathbf{V}^{\top}$, where max is elementwise. That is, the shrinkage operator shifts the singular values down, and truncates any negative values to zero. This step reduces the nuclear norm.

Even though the problem is convex, convergence can be slow. We follow [10] and use a continuation or homotopy method to improve the speed. This involves beginning with a large value $\mu_1 > \mu$ and solving a sequence of subproblems, each with a decreasing value and using the previous solution as its initial point. The sequence of values is determined by a decay parameter $\eta_\mu$: $\mu_{k+1} = \max\{\mu_k\eta_\mu, \mu\}, \quad k = 1, \ldots, L-1$, where $\mu$ is the final value to use, and $L$ is the number of rounds of continuation. The complete FPC algorithm for MC-b is listed in Algorithm 1.

A minor modification of the argument in [10] reveals that as long as we choose non-negative step sizes satisfying $\tau_{\mathbf{b}} < 4|\Omega_{\mathbf{Y}}|/(\lambda n)$ and $\tau_{\mathbf{Z}} < \min\{4|\Omega_{\mathbf{Y}}|/\lambda, |\Omega_{\mathbf{X}}|\}$, the algorithms MC-b will be guaranteed to converge to a global optimum. Indeed, to guarantee convergence, we only need that the gradient step is *non-expansive* in the sense that

$$\|\mathbf{b}_1 - \tau_{\mathbf{b}}g(\mathbf{b}_1) - \mathbf{b}_2 + \tau_{\mathbf{b}}g(\mathbf{b}_2)\|^2 + \|\mathbf{Z}_1 - \tau_{\mathbf{Z}}g(\mathbf{Z}_1) - \mathbf{Z}_2 + \tau_{\mathbf{Z}}g(\mathbf{Z}_2)\|_F^2 \leq \|\mathbf{b}_1 - \mathbf{b}_2\|^2 + \|\mathbf{Z}_1 - \mathbf{Z}_2\|_F^2$$

for all $\mathbf{b}_1, \mathbf{b}_2, \mathbf{Z}_1$, and $\mathbf{Z}_2$. Our choice of $\tau_{\mathbf{b}}$ and $\tau_{\mathbf{Z}}$ guarantee such non-expansiveness. Once this non-expansiveness is satisfied, the remainder of the convergence analysis is the same as in [10].

### 3.2 Fixed Point Continuation for MC-1

Our modified FPC method for MC-1 is similar except for two differences. First, there is no bias variable $\mathbf{b}$. Second, the shrinkage step will in general not satisfy the all-1-row constraints in (3). Thus, we add a third projection step at the end of each iteration to project $\mathbf{Z}^{k+1}$ back to the feasible region, by simply setting its last row to all 1's. The complete algorithm for MC-1 is given in Algorithm 2. We were unable to prove convergence for this gradient + shrinkage + projection algorithm. Nonetheless, in our empirical experiments, Algorithm 2 always converges and tends to outperform MC-b. The two algorithms have about the same convergence speed.

## 4 Experiments

We now empirically study the ability of matrix completion to perform multi-class transductive classification when there is missing data. We first present a family of 24 experiments on a synthetic task by systematically varying different aspects of the task, including the rank of the problem, noise level, number of items, and observed label and feature percentage. We then present experiments on two real-world datasets: music emotions and yeast microarray. In each experiments, we compare MC-b and MC-1 against four other baseline algorithms. Our results show that MC-1 consistently outperforms other methods, and MC-b follows closely.

**Parameter Tuning and Other Settings for MC-b and MC-1:** To tune the parameters $\mu$ and $\lambda$, we use 5-fold cross validation (CV) separately for each experiment. Specifically, we randomly

divide $\Omega_{\mathbf{X}}$ and $\Omega_{\mathbf{Y}}$ into five disjoint subsets each. We then run our matrix completion algorithms using $\frac{4}{5}$ of the observed entries, measure its performance on the remaining $\frac{1}{5}$, and average over the five folds. Since our main goal is to predict unobserved labels, we use label error as the CV performance criterion to select parameters. Note that tuning $\mu$ is quite efficient since all values under consideration can be evaluated in one run of the continuation method. We set $\eta_{\mu} = 0.25$ and, as in [10], consider $\mu$ values starting at $\sigma_1 \eta_{\mu}$, where $\sigma_1$ is the largest singular value of the matrix of observed entries in $[\mathbf{Y}; \mathbf{X}]$ (with the unobserved entries set to 0), and decrease $\mu$ until $10^{-5}$. The range of $\lambda$ values considered was $\{10^{-3}, 10^{-2}, 10^{-1}, 1\}$. We initialized $\mathbf{b}_0$ to be all zero and $\mathbf{Z}_0$ to be the rank-1 approximation of the matrix of observed entries in $[\mathbf{Y}; \mathbf{X}]$ (with unobserved entries set to 0) obtained by performing an SVD and reconstructing the matrix using only the largest singular value and corresponding left and right singular vectors. The step sizes were set as follows: $\tau_{\mathbf{Z}} = \min(\frac{3.8|\Omega_{\mathbf{Y}}|}{\lambda}, |\Omega_{\mathbf{X}}|)$, $\tau_{\mathbf{b}} = \frac{3.8|\Omega_{\mathbf{Y}}|}{\lambda n}$. Convergence was defined as relative change in objective functions (2)(3) smaller than $10^{-5}$.

**Baselines:** We compare to the following baselines, each consisting of some missing feature imputation step on $\mathbf{X}$ first, then using a standard SVM to predict the labels: **[FPC+SVM]** Matrix completion on $\mathbf{X}$ alone using FPC [10]. **[EM($k$)+SVM]** Expectation Maximization algorithm to impute missing $\mathbf{X}$ entries using a mixture of $k$ Gaussian components. As in [9], missing features, mixing component parameters, and the assignments of items to components are treated as hidden variables, which are estimated in an iterative manner to maximize the likelihood of the data. **[Mean+SVM]** Impute each missing feature by the mean of the observed entries for that feature. **[Zero+SVM]** Impute missing features by filling in zeros.

After imputation, an SVM is trained using the available (noisy) labels in $\Omega_{\mathbf{Y}}$ for that task, and predictions are made for the rest of the labels. All SVMs are linear, trained using SVMlin[2], and the regularization parameter is tuned using 5-fold cross validation separately for each task. The range of parameter values considered was $\{10^{-8}, 10^{-7}, \ldots, 10^7, 10^8\}$.

**Evaluation Method:** To evaluate performance, we consider two measures: *transductive label error*, i.e., the percentage of unobserved labels predicted incorrectly; and *relative feature imputation error* $\left( \sum_{ij \notin \Omega_{\mathbf{X}}} (x_{ij} - \hat{x}_{ij})^2 \right) / \sum_{ij \notin \Omega_{\mathbf{X}}} x_{ij}^2$, where $\hat{x}$ is the predicted feature value. In the tables below, for each parameter setting, we report the mean performance (and standard deviation in parenthesis) of different algorithms over 10 random trials. The best algorithm within each parameter setting, as well as any statistically indistinguishable algorithms via a two-tailed paired $t$-test at significance level $\alpha = 0.05$, are marked in bold.

### 4.1 Synthetic Data Experiments

**Synthetic Data Generation:** We generate a family of synthetic datasets to systematically explore the performance of the algorithms. We first create a rank-$r$ matrix $\mathbf{X^0} = \mathbf{L}\mathbf{R}^{\top}$, where $\mathbf{L} \in \mathbb{R}^{d \times r}$ and $\mathbf{R} \in \mathbb{R}^{n \times r}$ with entries drawn *iid* from $\mathcal{N}(0,1)$. We then normalize $\mathbf{X^0}$ such that its entries have variance 1. Next, we create a weight matrix $\mathbf{W} \in \mathbb{R}^{t \times d}$ and bias vector $\mathbf{b} \in \mathbb{R}^t$, with all entries drawn *iid* from $\mathcal{N}(0, 10)$. We then produce $\mathbf{X}, \mathbf{Y^0}, \mathbf{Y}$ according to section 2.1. Finally, we produce the random $\Omega_{\mathbf{X}}, \Omega_{\mathbf{Y}}$ masks with $\omega$ percent observed entries.

Using the above procedure, we vary $\omega = 10\%, 20\%, 40\%$, $n = 100, 400$, $r = 2, 4$, and $\sigma_{\epsilon}^2 = 0.01, 0.1$, while fixing $t = 10, d = 20$, to produce 24 different parameter settings. For each setting, we generate 10 trials, where the randomness is in the data and mask.

**Synthetic experiment results:** Table 1 shows the transductive label errors, and Table 2 shows the relative feature imputation errors, on the synthetic datasets. We make several observations.

Observation 1: MC-b and MC-1 are the best for feature imputation, as Table 2 shows. However, the imputations are not perfect, because in these particular parameter settings the ratio between the number of observed entries over the degrees of freedom needed to describe the feature matrix (i.e., $r(d + n - r)$) is below the necessary condition for perfect matrix completion [2], and because there is some feature noise. Furthermore, our CV tuning procedure selects parameters $\mu, \lambda$ to optimize label error, which often leads to suboptimal imputation performance. In a separate experiment (not reported here) when we made the ratio sufficiently large and without noise, and specifically tuned for

Table 1: Transductive label error of six algorithms on the 24 synthetic datasets. The varying parameters are feature noise $\sigma_\epsilon^2$, $\mathrm{rank}(\mathbf{X^0}) = r$, number of items $n$, and observed label and feature percentage $\omega$. Each row is for a unique parameter combination. Each cell shows the mean(standard deviation) of transductive label error (in percentage) over 10 random trials. The "meta-average" row is the simple average over all parameter settings and all trials.

| $\sigma_\epsilon^2$ | $r$ | $n$ | $\omega$ | MC-b | MC-1 | FPC+SVM | EM1+SVM | Mean+SVM | Zero+SVM |
|---|---|---|---|---|---|---|---|---|---|
| 0.01 | 2 | 100 | 10% | 37.8(4.0) | **31.8(4.3)** | **34.8(7.0)** | 34.6(3.9) | 40.5(5.7) | 40.5(5.1) |
| | | | 20% | 23.5(2.9) | **17.0(2.2)** | **17.6(2.1)** | 19.7(2.4) | 28.7(4.1) | 27.4(4.4) |
| | | | 40% | 15.1(3.1) | 10.8(1.8) | **9.6(1.5)** | 10.4(1.0) | 16.5(2.5) | 15.4(2.3) |
| | | 400 | 10% | 26.5(2.0) | **19.9(1.7)** | 23.7(1.7) | 24.2(1.9) | 32.4(2.9) | 31.5(2.7) |
| | | | 20% | 15.9(2.5) | **11.7(1.9)** | 12.6(2.2) | **12.0(1.9)** | 20.0(1.9) | 19.7(1.7) |
| | | | 40% | 11.7(2.0) | 8.0(1.6) | **7.2(1.8)** | **7.3(1.4)** | 12.2(1.8) | 12.1(2.0) |
| | 4 | 100 | 10% | 42.5(4.0) | **40.8(4.4)** | **41.5(2.6)** | **43.2(2.2)** | 43.5(2.9) | **42.9(2.9)** |
| | | | 20% | 33.2(2.3) | **26.2(2.8)** | **26.7(1.7)** | 30.8(2.7) | 35.5(1.4) | 33.9(1.5) |
| | | | 40% | 19.6(3.1) | **14.3(2.7)** | **13.6(2.6)** | **14.1(2.4)** | 22.5(2.0) | 21.7(2.3) |
| | | 400 | 10% | 35.3(3.1) | **32.1(1.6)** | **33.4(1.6)** | 34.2(1.8) | 37.7(1.2) | 38.2(1.4) |
| | | | 20% | 24.4(2.3) | **19.1(1.3)** | 20.5(1.4) | **19.8(1.1)** | 26.9(1.5) | 26.9(1.3) |
| | | | 40% | 14.6(1.8) | 9.5(0.5) | 9.2(0.9) | **8.6(1.1)** | 16.4(1.2) | 16.5(1.3) |
| 0.1 | 2 | 100 | 10% | 39.6(5.5) | **34.6(3.5)** | **37.3(6.4)** | 40.2(5.3) | 41.5(6.0) | 41.0(5.7) |
| | | | 20% | 25.2(2.6) | **20.1(1.7)** | **21.6(2.6)** | 26.8(3.7) | 31.8(4.7) | 29.9(4.0) |
| | | | 40% | 15.7(3.1) | **12.6(1.4)** | **13.2(2.0)** | 15.1(2.4) | 18.5(2.7) | 17.2(2.4) |
| | | 400 | 10% | 27.6(2.1) | **22.6(1.9)** | 27.6(2.4) | 28.8(2.6) | 34.5(3.3) | 33.6(2.8) |
| | | | 20% | 18.0(2.2) | **15.2(1.7)** | 16.8(2.3) | 18.4(2.5) | 22.6(2.4) | 21.8(2.5) |
| | | | 40% | 12.0(2.1) | **10.1(1.3)** | **10.4(2.1)** | 11.1(1.9) | 14.1(2.0) | 14.0(2.4) |
| | 4 | 100 | 10% | **42.5(4.3)** | 41.5(2.5) | **42.3(2.0)** | 45.6(1.9) | 44.6(2.9) | 43.6(2.3) |
| | | | 20% | 33.3(1.9) | **29.0(2.2)** | **30.9(3.1)** | 34.9(3.0) | 36.2(2.3) | 35.4(1.6) |
| | | | 40% | 21.4(2.7) | **18.4(3.1)** | **18.7(2.4)** | 21.6(2.4) | 23.9(2.0) | 23.3(2.5) |
| | | 400 | 10% | 36.3(2.7) | **34.0(1.7)** | **35.1(1.2)** | 36.3(1.4) | 38.7(1.3) | 39.1(1.2) |
| | | | 20% | 25.5(2.0) | **21.8(1.0)** | 23.8(1.5) | 25.1(1.4) | 28.4(1.7) | 28.4(1.8) |
| | | | 40% | 16.0(1.8) | **12.8(0.8)** | 13.9(1.2) | 14.7(1.3) | 18.3(1.2) | 18.2(1.2) |
| | | meta-average | | 25.6 | 21.4 | 22.6 | 24.1 | 28.6 | 28.0 |

imputation error, both MC-b and MC-1 did achieve perfect feature imputation. Also, FPC+SVM is slightly worse in feature imputation. This may seem curious as FPC focuses exclusively on imputing $\mathbf{X}$. We believe the fact that MC-b and MC-1 can use information in $\mathbf{Y}$ to enhance feature imputation in $\mathbf{X}$ made them better than FPC+SVM.

Observation 2: MC-1 is the best for multi-label transductive classification, as suggested by Table 1. Surprisingly, the feature imputation advantage of MC-b did not translate into classification, and FPC+SVM took second place.

Observation 3: The same factors that affect standard matrix completion also affect classification performance of MC-b and MC-1. As the tables show, everything else being equal, less feature noise (smaller $\sigma_\epsilon^2$), lower rank $r$, more items, or more observed features and labels, reduce label error. Beneficial combination of these factors (the $6^{th}$ row) produces the lowest label errors.

**Matrix completion benefits from more tasks.** We performed one additional synthetic data experiment examining the effect of $t$ (the number of tasks) on MC-b and MC-1, with the remaining data parameters fixed at $\omega = 10\%$, $n = 400$, $r = 2$, $d = 20$, and $\sigma_\epsilon^2 = 0.01$. Table 3 reveals that both MC methods achieve statistically significantly better label prediction and imputation performance with $t = 10$ than with only $t = 2$ (as determined by two-sample $t$-tests at significance level 0.05).

## 4.2 Music Emotions Data Experiments

In this task introduced by Trohidis *et al.* [14], the goal is to predict which of several types of emotion are present in a piece of music. The data[3] consists of $n = 593$ songs of a variety of musical genres, each labeled with one or more of $t = 6$ emotions (i.e., amazed-surprised, happy-pleased, relaxing-calm, quiet-still, sad-lonely, and angry-fearful). Each song is represented by $d = 72$ features (8 rhythmic, 64 timbre-based) automatically extracted from a 30-second sound clip.

Table 2: Relative feature imputation error on the synthetic datasets. The algorithm Zero+SVM is not shown because it by definition has relative feature imputation error 1.

| $\sigma_\epsilon^2$ | $r$ | $n$ | $\omega$ | MC-b | MC-1 | FPC+SVM | EM1+SVM | Mean+SVM |
|---|---|---|---|---|---|---|---|---|
| 0.01 | 2 | 100 | 10% | **0.84(0.04)** | **0.87(0.06)** | **0.88(0.06)** | 1.01(0.12) | 1.06(0.02) |
| | | | 20% | **0.54(0.08)** | **0.57(0.06)** | **0.57(0.07)** | 0.67(0.13) | 1.03(0.02) |
| | | | 40% | **0.29(0.06)** | **0.27(0.06)** | **0.27(0.06)** | 0.34(0.03) | 1.01(0.01) |
| | | 400 | 10% | **0.73(0.03)** | **0.72(0.04)** | 0.76(0.03) | 0.79(0.07) | 1.02(0.01) |
| | | | 20% | **0.43(0.04)** | 0.46(0.05) | 0.50(0.04) | **0.45(0.04)** | 1.01(0.00) |
| | | | 40% | 0.30(0.10) | **0.22(0.04)** | 0.24(0.05) | **0.21(0.04)** | 1.00(0.00) |
| | 4 | 100 | 10% | **0.99(0.04)** | **0.96(0.03)** | **0.96(0.03)** | 1.22(0.11) | 1.05(0.01) |
| | | | 20% | **0.77(0.05)** | **0.78(0.05)** | **0.77(0.04)** | 0.92(0.07) | 1.02(0.01) |
| | | | 40% | **0.42(0.07)** | **0.40(0.03)** | 0.42(0.04) | 0.49(0.04) | 1.01(0.01) |
| | | 400 | 10% | **0.87(0.04)** | **0.88(0.03)** | 0.89(0.01) | 1.00(0.08) | 1.01(0.00) |
| | | | 20% | **0.69(0.07)** | **0.67(0.04)** | 0.69(0.03) | **0.66(0.03)** | 1.01(0.00) |
| | | | 40% | 0.34(0.05) | 0.34(0.03) | 0.38(0.03) | **0.29(0.02)** | 1.00(0.00) |
| 0.1 | 2 | 100 | 10% | **0.92(0.05)** | **0.93(0.04)** | **0.93(0.05)** | 1.18(0.10) | 1.06(0.02) |
| | | | 20% | **0.69(0.07)** | 0.72(0.06) | 0.74(0.06) | 0.94(0.07) | 1.03(0.02) |
| | | | 40% | **0.51(0.05)** | **0.52(0.05)** | 0.53(0.05) | 0.67(0.08) | 1.02(0.01) |
| | | 400 | 10% | **0.79(0.03)** | 0.80(0.03) | 0.84(0.03) | 0.96(0.07) | 1.02(0.01) |
| | | | 20% | **0.64(0.06)** | **0.64(0.06)** | 0.67(0.04) | 0.73(0.07) | 1.01(0.00) |
| | | | 40% | 0.48(0.04) | **0.45(0.05)** | 0.49(0.05) | 0.57(0.07) | 1.00(0.00) |
| | 4 | 100 | 10% | **1.01(0.04)** | **0.97(0.03)** | **0.97(0.03)** | 1.25(0.05) | 1.05(0.02) |
| | | | 20% | **0.84(0.03)** | **0.85(0.03)** | **0.85(0.03)** | 1.07(0.06) | 1.02(0.01) |
| | | | 40% | **0.59(0.03)** | **0.61(0.04)** | 0.63(0.04) | 0.80(0.09) | 1.01(0.01) |
| | | 400 | 10% | **0.90(0.02)** | **0.92(0.02)** | 0.92(0.01) | 1.08(0.07) | 1.01(0.01) |
| | | | 20% | **0.75(0.04)** | **0.77(0.02)** | 0.79(0.03) | 0.86(0.05) | 1.01(0.00) |
| | | | 40% | 0.56(0.03) | **0.55(0.04)** | 0.59(0.04) | 0.66(0.06) | 1.00(0.00) |
| | | meta-average | | 0.66 | 0.66 | 0.68 | 0.78 | 1.02 |

Table 3: More tasks help matrix completion ($\omega = 10\%$, $n = 400$, $r = 2$, $d = 20$, $\sigma_\epsilon^2 = 0.01$).

| $t$ | MC-b | MC-1 | FPC+SVM | MC-b | MC-1 | FPC+SVM |
|---|---|---|---|---|---|---|
| 2 | 30.1(2.8) | 22.9(2.2) | 20.5(2.5) | 0.78(0.07) | 0.78(0.04) | 0.76(0.03) |
| 10 | **26.5(2.0)** | **19.9(1.7)** | 23.7(1.7) | **0.73(0.03)** | **0.72(0.04)** | 0.76(0.03) |
| | transductive label error | | | relative feature imputation error | | |

Table 4: Performance on the music emotions data.

| $\omega =40\%$ | 60% | 80% | Algorithm | $\omega =40\%$ | 60% | 80% |
|---|---|---|---|---|---|---|
| 28.0(1.2) | 25.2(1.0) | 22.2(1.6) | MC-b | 0.69(0.05) | 0.54(0.10) | 0.41(0.02) |
| 27.4(0.8) | **23.7(1.6)** | **19.8(2.4)** | MC-1 | 0.60(0.05) | 0.46(0.12) | 0.25(0.03) |
| 26.9(0.7) | 25.2(1.6) | 24.4(2.0) | FPC+SVM | 0.64(0.01) | 0.46(0.02) | 0.31(0.03) |
| **26.0(1.1)** | **23.6(1.1)** | 21.2(2.3) | EM1+SVM | 0.46(0.09) | 0.23(0.04) | **0.13(0.01)** |
| **26.2(0.9)** | **23.1(1.2)** | 21.6(1.6) | EM4+SVM | 0.49(0.10) | 0.27(0.04) | 0.15(0.02) |
| **26.3(0.8)** | 24.2(1.0) | 22.6(1.3) | Mean+SVM | **0.18(0.00)** | **0.19(0.00)** | 0.20(0.01) |
| 30.3(0.6) | 28.9(1.1) | 25.7(1.4) | Zero+SVM | 1.00(0.00) | 1.00(0.00) | 1.00(0.00) |
| transductive label error | | | | relative feature imputation error | | |

We vary the percentage of observed entries $\omega = 40\%, 60\%, 80\%$. For each $\omega$, we run 10 random trials with different masks $\Omega_{\mathbf{X}}, \Omega_{\mathbf{Y}}$. For this dataset, we tuned only $\mu$ with CV, and set $\lambda = 1$.

The results are in Table 4. Most importantly, these results show that MC-1 is useful for this real-world multi-label classification problem, leading to the best (or statistically indistinguishable from the best) transductive error performance with 60% and 80% of the data available, and close to the best with only 40%.

We also compared these algorithms against an "oracle baseline" (not shown in the table). In this baseline, we give 100% features (i.e., no indices are missing from $\Omega_{\mathbf{X}}$) and the training labels in $\Omega_{\mathbf{Y}}$ to a standard SVM, and let it predict the unspecified labels. On the same random trials, for observed percentage $\omega = 40\%, 60\%, 80\%$, the oracle baseline achieved label error rate $22.1(0.8), 21.3(0.8), 20.5(1.8)$ respectively. Interestingly, MC-1 with $\omega = 80\%$ (19.8) is statistically indistinguishable from the oracle baseline.

### 4.3 Yeast Microarray Data Experiments

This dataset comes from a biological domain and involves the problem of Yeast gene functional classification. We use the data studied by Elisseeff and Weston [5], which contains $n = 2417$ examples (Yeast genes) with $d = 103$ input features (results from microarray experiments).[4] We follow the approach of [5] and predict each gene's membership in $t = 14$ functional classes. For this larger dataset, we omitted the computationally expensive EM4+SVM methods, and tuned only $\mu$ for matrix completion while fixing $\lambda = 1$.

Table 5 reveals that MC-b leads to statistically significantly lower transductive label error for this biological dataset. Although not highlighted in the table, MC-1 is also statistically better than the SVM methods in label error. In terms of feature imputation performance, the MC methods are weaker than FPC+SVM. However, it seems simultaneously predicting the missing labels and features appears to provide a large advantage to the MC methods. It should be pointed out that all algorithms except Zero+SVM in fact have small but non-zero standard deviation on imputation error, despite what the fixed-point formatting in the table suggests. For instance, with $\omega = 40\%$, the standard deviation is 0.0009 for MC-1, 0.0011 for FPC+SVM, and 0.0001 for Mean+SVM.

Again, we compared these algorithms to an oracle SVM baseline with 100% observed entries in $\Omega_{\mathbf{X}}$. The oracle SVM approach achieves label error of 20.9(0.1), 20.4(0.2), and 20.1(0.3) for $\omega =$40%, 60%, and 80% observed labels, respectively. Both MC-b and MC-1 significantly outperform this oracle under paired $t$-tests at significance level 0.05. We attribute this advantage to a combination of multi-label learning and transduction that is intrinsic to our matrix completion methods.

Table 5: Performance on the yeast data.

| $\omega =$40% | 60% | 80% | Algorithm | $\omega =$40% | 60% | 80% |
|---|---|---|---|---|---|---|
| **16.1(0.3)** | **12.2(0.3)** | **8.7(0.4)** | MC-b | 0.83(0.02) | 0.76(0.00) | 0.73(0.02) |
| 16.7(0.3) | 13.0(0.2) | **8.5(0.4)** | MC-1 | 0.86(0.00) | 0.92(0.00) | 0.74(0.00) |
| 21.5(0.3) | 20.8(0.3) | 20.3(0.3) | FPC+SVM | **0.81(0.00)** | **0.76(0.00)** | **0.72(0.00)** |
| 22.0(0.2) | 21.2(0.2) | 20.4(0.2) | EM1+SVM | 1.15(0.02) | 1.04(0.02) | 0.77(0.01) |
| 21.7(0.2) | 21.1(0.2) | 20.5(0.4) | Mean+SVM | 1.00(0.00) | 1.00(0.00) | 1.00(0.00) |
| 21.6(0.2) | 21.1(0.2) | 20.5(0.4) | Zero+SVM | 1.00(0.00) | 1.00(0.00) | 1.00(0.00) |

transductive label error        relative feature imputation error

## 5 Discussions and Future Work

We have introduced two matrix completion methods for multi-label transductive learning with missing features, which outperformed several baselines. In terms of problem formulation, our methods differ considerably from sparse multi-task learning [11, 1, 13] in that we regularize the feature and label matrix directly, without ever learning explicit weight vectors. Our methods also differ from multi-label prediction via reduction to binary classification or ranking [15], and via compressed sensing [7], which assumes sparsity in that each item has a small number of positive labels, rather than the low-rank nature of feature matrices. These methods do not naturally allow for missing features. Yet other multi-label methods identify a subspace of highly predictive features across tasks in a first stage, and learn in this subspace in a second stage [8, 12]. Our methods do not require separate stages. Learning in the presence of missing data typically involves imputation followed by learning with completed data [9]. Our methods perform imputation plus learning in one step, similar to EM on missing labels and features [6], but the underlying model assumption is quite different.

A drawback of our methods is their restriction to linear classifiers only. One future extension is to explicitly map the partial feature matrix to a partially observed polynomial (or other) kernel Gram matrix, and apply our methods there. Though such mapping proliferates the missing entries, we hope that the low-rank structure in the kernel matrix will allow us to recover labels that are nonlinear functions of the original features.

**Acknowledgements:** This work is supported in part by NSF IIS-0916038, NSF IIS-0953219, AFOSR FA9550-09-1-0313, and AFOSR A9550-09-1-0423. We also wish to thank Brian Eriksson for useful discussions and source code implementing EM-based imputation.

## Footnotes

[1]While the primary method of [10] is Fixed Point Continuation with Approximate Singular Value Decomposition (FPCA), where the approximate SVD is used to speed up the algorithm, we opt to use an exact SVD for simplicity and will refer to the method simply as FPC.

[2]http://vikas.sindhwani.org/svmlin.html

[3]Available at http://mulan.sourceforge.net/datasets.html

[4]Available at http://mulan.sourceforge.net/datasets.html

# References

[1] Andreas Argyriou, Charles A. Micchelli, and Massimiliano Pontil. On spectral learning. *Journal of Machine Learning Research*, 11:935–953, 2010.

[2] Emmanuel J. Candès and Benjamin Recht. Exact matrix completion via convex optimization. *Foundations of Computational Mathematics*, 9:717–772, 2009.

[3] Emmanuel J. Candès and Terence Tao. The power of convex relaxation: Near-optimal matrix completion. *IEEE Transactions on Information Theory*, 56:2053–2080, 2010.

[4] Olivier Chapelle, Alexander Zien, and Bernhard Schölkopf, editors. *Semi-supervised learning*. MIT Press, 2006.

[5] André Elisseeff and Jason Weston. A kernel method for multi-labelled classification. In Thomas G. Dietterich, Suzanna Becker, and Zoubin Ghahramani, editors, *NIPS*, pages 681–687. MIT Press, 2001.

[6] Zoubin Ghahramani and Michael I. Jordan. Supervised learning from incomplete data via an EM approach. In *Advances in Neural Information Processing Systems 6*, pages 120–127. Morgan Kaufmann, 1994.

[7] Daniel Hsu, Sham Kakade, John Langford, and Tong Zhang. Multi-label prediction via compressed sensing. In Y. Bengio, D. Schuurmans, J. Lafferty, C. K. I. Williams, and A. Culotta, editors, *Advances in Neural Information Processing Systems 22*, pages 772–780. 2009.

[8] Shuiwang Ji, Lei Tang, Shipeng Yu, and Jieping Ye. Extracting shared subspace for multi-label classification. In *KDD '08: Proceeding of the 14th ACM SIGKDD international conference on Knowledge discovery and data mining*, pages 381–389, New York, NY, USA, 2008. ACM.

[9] Roderick J. A. Little and Donald B. Rubin. *Statistical Analysis with Missing Data*. Wiley-Interscience, 2nd edition, September 2002.

[10] Shiqian Ma, Donald Goldfarb, and Lifeng Chen. Fixed point and Bregman iterative methods for matrix rank minimization. *Mathematical Programming Series A*, to appear (published online September 23, 2009).

[11] Guillaume Obozinski, Ben Taskar, and Michael I. Jordan. Joint covariate selection and joint subspace selection for multiple classification problems. *Statistics and Computing*, 20(2):231–252, 2010.

[12] Piyush Rai and Hal Daume. Multi-label prediction via sparse infinite CCA. In Y. Bengio, D. Schuurmans, J. Lafferty, C. K. I. Williams, and A. Culotta, editors, *Advances in Neural Information Processing Systems 22*, pages 1518–1526. 2009.

[13] Nathan Srebro and Adi Shraibman. Rank, trace-norm and max-norm. In *Proceedings of the 18th Annual Conference on Learning Theory*, pages 545–560. Springer-Verlag, 2005.

[14] K. Trohidis, G. Tsoumakas, G. Kalliris, and I. Vlahavas. Multilabel classification of music into emotions. In *Proc. 9th International Conference on Music Information Retrieval (ISMIR 2008), Philadelphia, PA, USA, 2008*, 2008.

[15] G. Tsoumakas, I. Katakis, and I. Vlahavas. Mining multi-label data. In *Data Mining and Knowledge Discovery Handbook*. Springer, 2nd edition, 2010.

[16] Xiaojin Zhu and Andrew B. Goldberg. *Introduction to Semi-Supervised Learning*. Morgan & Claypool, 2009.

